# Ensemble Learning and Linear Response Theory for ICA

**Pedro A.d.F.R. Højen-Sørensen[1], Ole Winther[2] , Lars Kai Hansen[1]**
[1]Department of Mathematical Modelling, Technical University of Denmark B321
DK-2800 Lyngby, Denmark, `phs,lkhansen@imm.dtu.dk`
[2]Theoretical Physics, Lund University, Sölvegatan 14 A
S-223 62 Lund, Sweden, `winther@nimis.thep.lu.se`

## Abstract

We propose a general Bayesian framework for performing independent component analysis (ICA) which relies on ensemble learning and linear response theory known from statistical physics. We apply it to both discrete and continuous sources. For the continuous source the underdetermined (overcomplete) case is studied. The naive mean-field approach fails in this case whereas linear response theory–which gives an improved estimate of covariances–is very efficient. The examples given are for sources without temporal correlations. However, this derivation can easily be extended to treat temporal correlations. Finally, the framework offers a simple way of generating new ICA algorithms without needing to define the prior distribution of the sources explicitly.

## 1   Introduction

Reconstruction of statistically independent source signals from linear mixtures is an active research field. For historical background and early references see e.g. [1]. The source separation problem has a Bayesian formulation, see e.g., [2, 3] for which there has been some recent progress based on ensemble learning [4].

In the Bayesian framework, the covariances of the sources are needed in order to estimate the mixing matrix and the noise level. Unfortunately, ensemble learning using factorized trial distributions only treats self-interactions correctly and trivially predicts: $\langle S_i S_i \rangle - \langle S_i \rangle \langle S_j \rangle = 0$ for $i \neq j$. This naive mean-field (NMF) approximation first introduced in the neural computing context by Ref. [5] for Boltzmann machine learning may completely fail in some cases [6]. Recently, Kappen and Rodríguez [6] introduced an efficient learning algorithm for Boltzmann Machines based on linear response (LR) theory. LR theory gives a recipe for computing an improved approximation to the covariances directly from the solution to the NMF equations [7].

Ensemble learning has been applied in many contexts within neural computation, e.g. for sigmoid belief networks [8], where advanced mean field methods such as LR theory or TAP [9] may also be applicable. In this paper, we show how LR theory can be applied to independent component analysis (ICA). The performance of this approach is compared to the NMF approach. We observe that NMF may fail for high noise levels and binary

sources and for the underdetermined continuous case. In these cases the NMF approach ignores one of the sources and consequently overestimates the noise. The LR approach on the other hand succeeds in all cases studied.

The derivation of the mean-field equations are kept completely general and are thus valid for a general source prior (without temporal correlations). The final eqs. show that the mean-field framework may be used to propose ICA algorithms for which the source prior is only defined implicitly.

## 2 Probabilistic ICA

Following Ref. [10], we consider a collection of $N$ temporal measurements, $\mathbf{X} = \{X_{dt}\}$, where $X_{dt}$ denotes the measurement at the $d$th sensor at time $t$. Similarly, let $\mathbf{S} = \{S_{mt}\}$ denote a collection of $M$ mutually independent sources where $S_{m\cdot}$ is the $m$th source which in general may have temporal correlations. The measured signals $\mathbf{X}$ are assumed to be an instantaneous linear mixing of the sources corrupted with additive Gaussian noise $\mathbf{\Gamma}$, that is,

$$\mathbf{X} = \mathbf{AS} + \mathbf{\Gamma} \,, \tag{1}$$

where $\mathbf{A}$ is the mixing matrix. Furthermore, to simplify this exposition the noise is assumed to be iid Gaussian with variance $\sigma^2$. The likelihood of the parameters is then given by,

$$P(\mathbf{X}|\mathbf{A}, \sigma^2) = \int d\mathbf{S} P(\mathbf{X}|\mathbf{A}, \sigma^2, \mathbf{S}) \, P(\mathbf{S}) \,, \tag{2}$$

where $P(\mathbf{S})$ is the prior on the sources which might include temporal correlations. We will, however, throughout this paper assume that the sources are temporally uncorrelated. We choose to estimate the mixing matrix $\mathbf{A}$ and noise level $\sigma^2$ by Maximum Likelihood (ML-II). The saddlepoint of $P(\mathbf{X}|\mathbf{A}, \sigma^2)$ is attained at,

$$\frac{\partial \log P(\mathbf{X}|\mathbf{A}, \sigma^2)}{\partial \mathbf{A}} = 0 \quad : \quad \mathbf{A} = \mathbf{X}\langle \mathbf{S}\rangle^T \langle \mathbf{SS}^T\rangle^{-1} \tag{3}$$

$$\frac{\partial \log P(\mathbf{X}|\mathbf{A}, \sigma^2)}{\partial \sigma^2} = 0 \quad : \quad \sigma^2 = \frac{1}{DN}\langle \mathrm{Tr}(\mathbf{X} - \mathbf{AS})^T(\mathbf{X} - \mathbf{AS})\rangle \,, \tag{4}$$

where $\langle \cdot \rangle$ denotes an average over the posterior and $D$ is the number of sensors.

## 3 Mean field theory

First, we derive mean field equations using ensemble learning. Secondly, using linear response theory, we obtain improved estimates of the off-diagonal terms of $\langle \mathbf{SS}^T \rangle$ which are needed for estimating $\mathbf{A}$ and $\sigma^2$. The following derivation is performed for an arbitrary source prior.

### 3.1 Ensemble learning

We adopt a standard ensemble learning approach and approximate

$$P(\mathbf{S}|\mathbf{X}, \mathbf{A}, \sigma^2) = \frac{P(\mathbf{X}|\mathbf{A}, \sigma^2, \mathbf{S})P(\mathbf{S})}{P(\mathbf{X}|\mathbf{A}, \sigma^2)} \tag{5}$$

in a family of product distributions $Q(\mathbf{S}) = \prod_{mt} Q(S_{mt})$. It has been shown in Ref. [11] that for a Gaussian $P(\mathbf{X}|\mathbf{A}, \sigma^2, \mathbf{S})$, the optimal choice of $Q(S_{mt})$ is given by a Gaussian times the prior:

$$Q(S_{mt}) = \frac{P(S_{mt})e^{\frac{1}{2}\lambda_{mt}S_{mt}^2 + \gamma_{mt}S_{mt}}}{\int dS P(S)e^{\frac{1}{2}\lambda_{mt}S^2 + \gamma_{mt}S}} \,. \tag{6}$$

In the following, it is convenient to use standard physics notation to keep everything as general as possible. We therefore parameterize the Gaussian as,

$$P(\mathbf{X}|\mathbf{A},\sigma^2,\mathbf{S}) = P(\mathbf{X}|\mathbf{J},\mathbf{h},\mathbf{S}) = Ce^{\frac{1}{2}\operatorname{Tr}(\mathbf{S}^T\mathbf{J}\mathbf{S})+\operatorname{Tr}(\mathbf{h}^T\mathbf{S})}\ , \tag{7}$$

where $\mathbf{J} = -\mathbf{A}^T\mathbf{A}/\sigma^2$ is the $M \times M$ interaction matrix and $\mathbf{h} = \mathbf{A}^T\mathbf{X}/\sigma^2$ has the same dimensions as the source matrix $\mathbf{S}$. Note that $\mathbf{h}$ acts as an external field from which we can obtain all moments of the sources. This is a property that we will make use of in the next section when we derive the linear response corrections. The Kullback-Leibler divergence between the optimal product distribution $Q(\mathbf{S})$ and the true source posterior is given by

$$KL \quad = \quad \int d\mathbf{S}Q(\mathbf{S})\ln\frac{Q(\mathbf{S})}{P(\mathbf{S}|\mathbf{X},\mathbf{A},\sigma^2)} = \ln P(\mathbf{X}|\mathbf{A},\sigma^2) - \ln \hat{P}(\mathbf{X}|\mathbf{A},\sigma^2) \tag{8}$$

$$\ln \hat{P}(\mathbf{X}|\mathbf{A},\sigma^2) \quad = \quad \sum_{mt}\log\int dSP(S)e^{\frac{1}{2}\lambda_{mt}S^2+\gamma_{mt}S} + \frac{1}{2}\sum_{mt}(J_{mm}-\lambda_{mt})\langle S_{mt}^2\rangle$$

$$+\frac{1}{2}\operatorname{Tr}\langle\mathbf{S}^T\rangle(\mathbf{J}-\operatorname{diag}(\mathbf{J})\langle\mathbf{S}\rangle + \operatorname{Tr}(\mathbf{h}-\boldsymbol{\gamma})^T\langle\mathbf{S}\rangle + \ln C\ , \tag{9}$$

where $\hat{P}(\mathbf{X}|\mathbf{A},\sigma^2)$ is the naive mean field approximation to the Likelihood and $\operatorname{diag}(\mathbf{J})$ is the diagonal matrix of $\mathbf{J}$. The saddlepoints define the mean field equations;

$$\frac{\partial KL}{\partial\langle\mathbf{S}\rangle} = 0 \quad : \quad \boldsymbol{\gamma} = \mathbf{h} + (\mathbf{J}-\operatorname{diag}(\mathbf{J}))\langle\mathbf{S}\rangle \tag{10}$$

$$\frac{\partial KL}{\partial\langle S_{mt}^2\rangle} = 0 \quad : \quad \lambda_{mt} = J_{mm}\ . \tag{11}$$

The remaining two equations depend explicitly on the source prior, $P(\mathbf{S})$;

$$\frac{\partial KL}{\partial\gamma_{mt}} = 0 \quad : \quad \langle S_{mt}\rangle = \frac{\partial}{\partial\gamma_{mt}}\log\int dS_{mt}P(S_{mt})e^{\frac{1}{2}\lambda_{mt}S_{mt}^2+\gamma_{mt}S_{mt}}$$

$$\equiv f(\gamma_{mt},\lambda_{mt}) \tag{12}$$

$$\frac{\partial KL}{\partial\lambda_{mt}} = 0 \quad : \quad \langle S_{mt}^2\rangle = 2\frac{\partial}{\partial\lambda_{mt}}\log\int dS_{mt}P(S_{mt})e^{\frac{1}{2}\lambda_{mt}S_{mt}^2+\gamma_{mt}S_{mt}}\ . \tag{13}$$

In section 4, we calculate $f(\gamma_{mt},\lambda_{mt})$ for some of the prior distributions found in the ICA literature.

### 3.2 Linear response theory

As mentioned already, $\mathbf{h}$ acts as an external field. This makes it possible to calculate the means and covariances as derivatives of $\log P(\mathbf{X}|\mathbf{J},\mathbf{h})$, i.e.

$$\langle S_{mt}\rangle = \frac{\partial\log P(\mathbf{X}|\mathbf{J},\mathbf{h})}{\partial h_{mt}} \tag{14}$$

$$\chi_{mm'}^{tt'} \equiv \langle S_{mt}S_{m't'}\rangle - \langle S_{mt}\rangle\langle S_{m't'}\rangle = \frac{\partial^2\log P(\mathbf{X}|\mathbf{J},\mathbf{h})}{\partial h_{m't'}\partial h_{mt}} = \frac{\partial\langle S_{mt}\rangle}{\partial h_{m't'}}\ . \tag{15}$$

To derive an equation for $\chi_{mm'}^{tt'}$, we use eqs. (10), (11) and (12) to get

$$\chi_{mm'}^{tt'} \quad = \quad \frac{\partial f(\gamma_{mt},\lambda_{mt})}{\partial\gamma_{mt}}\frac{\partial\gamma_{mt}}{\partial h_{m't'}}$$

$$= \quad \frac{\partial f(\gamma_{mt},\lambda_{mt})}{\partial\gamma_{mt}}\left(\sum_{m'',m''\neq m}J_{mm''}\chi_{m''m'}^{tt} + \delta_{mm'}\right)\delta_{tt'}\ . \tag{16}$$

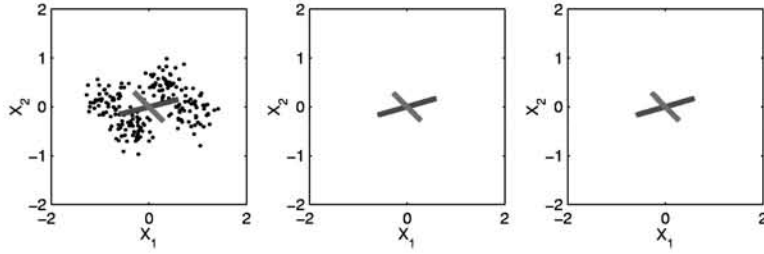

Figure 1: Binary source recovery for low noise level ($M = 2$, $D = 2$). Shows from left to right: +/- the column vectors of; the true **A** (with the observations superimposed); the estimated **A** (NMF); estimated **A** (LR).

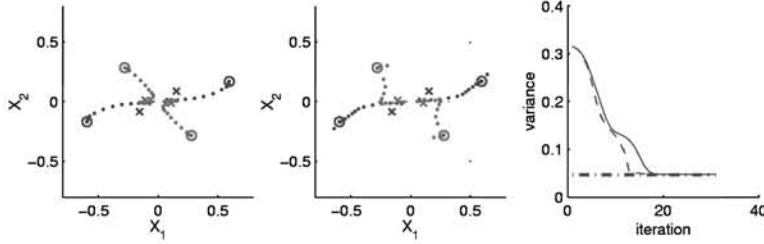

Figure 2: Binary source recovery for low noise level ($M = 2$, $D = 2$). Shows the dynamics of the fix-point iterations. From left to right; +/- the column vectors of **A** (NMF); +/- the column vectors of **A** (LR); variance $\sigma^2$ (solid:NMF, dashed:LR, thick dash-dotted: the true empirical noise variance).

We now see that the $\chi$-matrix factorizes in time $\chi_{mm'}^{tt'} = \delta_{tt'} \chi_{mm'}^{t}$. This is a direct consequence of the fact that the model has no temporal correlations. The above equation is linear and may straightforwardly be solved to yield

$$\chi_{mm'}^{t} = \left[ (\mathbf{\Lambda}^t - \mathbf{J})^{-1} \right]_{mm'} , \qquad (17)$$

where we have defined the diagonal matrix

$$\mathbf{\Lambda}^t = \mathrm{diag} \left( \frac{1}{\frac{\partial f(\gamma_{1t}, \lambda_{1t})}{\partial \gamma_{1t}}} + J_{11}, \ldots, \frac{1}{\frac{\partial f(\gamma_{Mt}, \lambda_{Mt})}{\partial \gamma_{Mt}}} + J_{MM} \right) .$$

At this point is appropriate to explain why linear response theory is more precise than using the factorized distribution which predicts $\chi_{mm'}^{t} = 0$ for non-diagonal terms. Here, we give an argument that can be found in Parisi's book on statistical field theory [7]: Let us assume that the approximate and exact distribution is close in some sense, i.e. $Q(\mathbf{S}) - P(\mathbf{S}|\mathbf{X}, \mathbf{A}, \sigma^2) = \varepsilon$ then $\langle S_{mt} S_{m't} \rangle_{\mathrm{ex}} = \langle S_{mt} S_{m't} \rangle_{\mathrm{ap}} + \mathcal{O}(\varepsilon)$. Mean field theory gives a lower bound on the log-Likelihood since $KL$, eq. (8) is non-negative. Consequently, the linear term vanishes in the expansion of the log-Likelihood: $\log P(\mathbf{X}|\mathbf{A}, \sigma^2) = \log \hat{P}(\mathbf{X}|\mathbf{A}, \sigma^2) + \mathcal{O}(\varepsilon^2)$. It is therefore more precise to obtain moments of the variables through derivatives of the approximate log-Likelihood, i.e. by linear response.

A final remark to complete the picture: if $\mathrm{diag}(\mathbf{J})$ in equation eq. (10) is exchanged with $\boldsymbol{\lambda}^t = \mathrm{diag}(\lambda_{1t}, \ldots, \lambda_{Mt})$ and likewise in the definition of $\mathbf{\Lambda}^t$ above we get TAP equations [9]. The TAP equation for $\lambda_{mt}$ is $\chi_{mm}^{t} = \frac{\partial f(\gamma_{mt}, \lambda_{mt})}{\partial \gamma_{mt}} = \left[ (\mathbf{\Lambda}^t - \mathbf{J})^{-1} \right]_{mm}$.

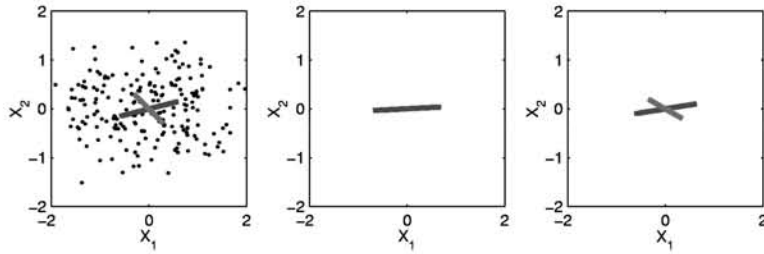

Figure 3: Binary source recovery for high noise level ($M = 2$, $D = 2$). Shows from left to right: +/- the column vectors of; the true $\mathbf{A}$ (with the observations superimposed); the estimated $\mathbf{A}$ (NMF); estimated $\mathbf{A}$ (LR).

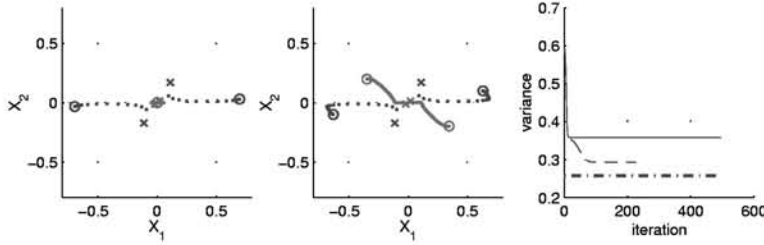

Figure 4: Binary source recovery for high noise level ($M = 2$, $D = 2$). Same plot as in figure 2.

## 4  Examples

In this section we compare the LR approach and the NMF approach on the noisy ICA model. The two approaches are demonstrated using binary and continous sources.

### 4.1  Binary source

Independent component analysis of binary sources (e.g. studied in [12]) is considered for data transmission using binary modulation schemes such as MSK or biphase (Manchester) codes. Here, we consider a binary source $S_{mt} \in \{-1, 1\}$ with prior distribution $P(S_{mt}) = \frac{1}{2}[\delta(S_{mt} - 1) + \delta(S_{mt} + 1)]$. In this case we get the well known mean field equations $\langle S_{mt} \rangle = \tanh(\gamma_{mt})$. Figures 1 and 2 show the results of the NMF approach as well as LR approach in a low-noise variance setting using two sources ($M = 2$) and two sensors ($D = 2$). Figures 3 and 4 show the same but in a high-noise setting. The dynamical plots show the trajectory of the fix-point iteration where 'x' marks the starting point and 'o' the final point. Ideally, the noise-less measurements would consist of the four combinations (with signs) of the columns in the mixing matrix. However, due to the noise, the measurement will be scattered around these "prototype" observations.

In the low-noise level setting both approaches find good approximations to the true mixing matrix and sources. However, the convergence rate of the LR approach is found to be faster. For high-noise variance the NMF approach fails to recover the true statistics. It is seen that one of the directions in the mixing matrix vanishes which in turn results in overestimating the noise variance.

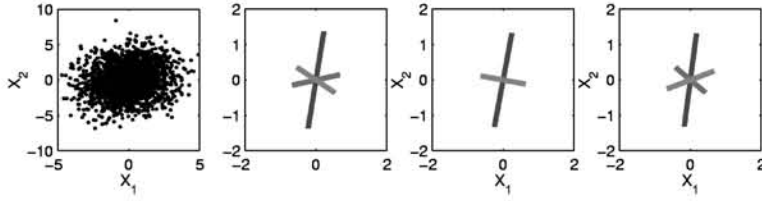

Figure 5: Overcomplete continuous source recovery with $M = 3$ and $D = 2$. Shows from left to right: the observations, +/- the column vectors of; the true **A**; the estimated **A** (NMF); estimated **A** (LR).

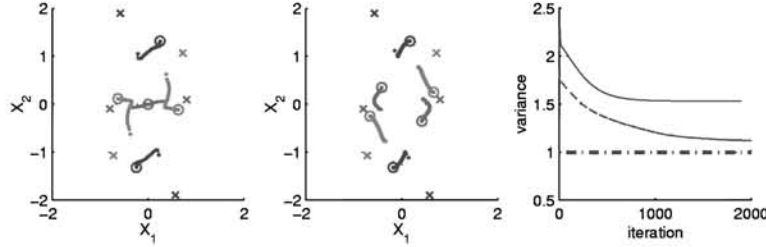

Figure 6: Overcomplete continuous source recovery with $M = 3$ and $D = 2$. Same plot as in figure 2. Note that the initial iteration step for **A** is very large.

## 4.2 Continuous Source

To give a tractable example which illustrates the improvement by LR, we consider the Gaussian prior $P(S_{mt}) \propto \exp(-\alpha S_{mt}^2/2)$ (not suitable for source separation). This leads to $f(\gamma_{mt}, \lambda_{mt}) = \gamma_{mt}/(\alpha - \lambda_{mt})$. Since we have a factorized distribution, ensemble learning predicts $\langle S_{mt} S_{m't'} \rangle - \langle S_{mt} \rangle \langle S_{m't'} \rangle = \delta_{mm'} \delta_{tt'} (\alpha - \lambda_{mt})^{-1} = \delta_{mm'} \delta_{tt'} (\alpha - J_{mm})^{-1}$, where the second equality follows from eq. (11). Linear response eq. (17) gives $\langle S_{mt} S_{m't'} \rangle - \langle S_{mt} \rangle \langle S_{m't'} \rangle = \delta_{tt'} \left[ (\alpha \mathbf{I} - \mathbf{J})^{-1} \right]_{mm'}$ which is identical with the exact result obtained by direct integration.

For the popular choice of prior $P(S_{mt}) = \frac{1}{\pi \cosh S_{mt}}$ [1], it is not possible to derive $f(\gamma_{mt}, \lambda_{mt})$ analytically. However, $f(\gamma_{mt}, \lambda_{mt})$ can be calculated analytically for the very similar Laplace distribution. Both these examples have positive kurtosis.

Mean field equations for negative kurtosis can be obtained using the prior $P(S_{mt}) \propto = \exp(-(S_{mt} - \mu)^2/2) + \exp(-(S_{mt} + \mu)^2/2)$ [1] leading to

$$\langle S_{mt} \rangle = \frac{1}{1 - \lambda_{mt}} \left( \gamma_{mt} + \mu \tanh \left( \frac{\mu \gamma_{mt}}{1 - \lambda_{mt}} \right) \right) .$$

Figure 5 and 6 show simulations using this source prior with $\mu = 1$ in an overcomplete setting with $D = 2$ and $M = 3$. Note that $\mu = 1$ yields a unimodal source distribution and hence qualitatively different from the bimodal prior considered in the binary case. In the overcomplete setting the NMF approach fails to recover the true sources. See [13] for further discussion of the overcomplete case.

# 5 Conclusion

We have presented a general ICA mean field framework based upon ensemble learning and linear response theory. The naive mean-field approach (pure ensemble learning) fails in some cases and we speculate that it is incapable of handling the overcomplete case (more sources than sensors). Linear response theory, on the other hand, succeeds in all the examples studied.

There are two directions in which we plan to extend this work: (1) to sources with temporal correlations and (2) to source models defined not by a parametric source prior, but directly in terms of the function $f$, which defines the mean field equations. Starting directly from the $f$-function makes it possible to test a whole range of implicitly defined source priors. A detailed analysis of a large selection of constrained and unconstrained source priors as well as comparisons of LR and the TAP approach can be found in [14].

### Acknowledgments

PHS wishes to thank Mike Jordan for stimulating discussions on the mean field and variational methods. This research is supported by the Swedish Foundation for Strategic Research as well as the Danish Research Councils through the Computational Neural Network Center (CONNECT) and the THOR Center for Neuroinformatics.

## References

[1] T.-W. Lee: *Independent Component Analysis*, Kluwer Academic Publishers, Boston (1998).

[2] A. Belouchrani and J.-F. Cardoso: *Maximum Likelihood Source Separation by the Expectation-Maximization Technique: Deterministic and Stochastic Implementation* In Proc. NOLTA, 49–53 (1995).

[3] D. MacKay: *Maximum Likelihood and Covariant Algorithms for Independent Components Analysis*. "Draft 3.7" (1996).

[4] H. Lappalainen and J.W. Miskin: *Ensemble Learning*, Advances in Independent Component Analysis, Ed. M. Girolami, In press (2000).

[5] C. Peterson and J. Anderson: *A Mean Field Theory Learning Algorithm for Neural Networks*, Complex Systems **1**, 995–1019 (1987).

[6] H. J. Kappen and F. B. Rodríguez: *Efficient Learning in Boltzmann Machines Using Linear Response Theory*, Neural Computation **10**, 1137–1156 (1998).

[7] G. Parisi: *Statistical Field Theory*, Addison Wesley, Reading Massachusetts (1988).

[8] L. K. Saul, T. Jaakkola and M. I. Jordan: *Mean Field Theory of Sigmoid Belief Networks*, Journal of Artificial Intelligence Research **4**, 61–76 (1996).

[9] M. Opper and O. Winther: *Tractable Approximations for Probabilistic Models: The Adaptive TAP Mean Field Approach*, Submitted to Phys. Rev. Lett. (2000).

[10] L. K. Hansen: *Blind Separation of Noisy Image Mixtures*, Advances in Independent Component Analysis, Ed. M. Girolami, In press (2000).

[11] L. Csató, E. Fokoué, M. Opper, B. Schottky and O. Winther: *Efficient Approaches to Gaussian Process Classification*, in Advances in Neural Information Processing Systems 12 (NIPS'99), Eds. S. A. Solla, T. K. Leen, and K.-R. Müller, MIT Press (2000).

[12] A.-J. van der Veen: *Analytical Method for Blind Binary Signal Separation* IEEE Trans. on Signal Processing 45(4) 1078–1082 (1997).

[13] M. S. Lewicki and T. J. Sejnowski: *Learning Overcomplete Representations*, Neural Computation **12**, 337–365 (2000).

[14] P. A. d. F. R. Højen-Sørensen, O. Winther and L. K. Hansen: *Mean Field Approaches to Independent Component Analysis*, In preparation.
